# Stable Dual Dynamic Programming

**Tao Wang**[*]   **Daniel Lizotte**   **Michael Bowling**   **Dale Schuurmans**
Department of Computing Science
University of Alberta
{trysi,dlizotte,bowling,dale}@cs.ualberta.ca

## Abstract

Recently, we have introduced a novel approach to dynamic programming and reinforcement learning that is based on maintaining explicit representations of stationary distributions instead of value functions. In this paper, we investigate the convergence properties of these dual algorithms both theoretically and empirically, and show how they can be scaled up by incorporating function approximation.

## 1   Introduction

Value function representations are dominant in algorithms for dynamic programming (DP) and reinforcement learning (RL). However, linear programming (LP) methods clearly demonstrate that the value function is not a necessary concept for solving sequential decision making problems. In LP methods, value functions only correspond to the primal formulation of the problem, while in the dual they are replaced by the notion of state (or state-action) *visit distributions* [1, 2, 3]. Despite the well known LP duality, dual representations have not been widely explored in DP and RL. Recently, we have showed that it is entirely possible to solve DP and RL problems in the dual representation [4]. Unfortunately, [4] did not analyze the convergence properties nor implement the proposed ideas. In this paper, we investigate the convergence properties of these newly proposed dual solution techniques, and show how they can be scaled up by incorporating function approximation. The proof techniques we use to analyze convergence are simple, but lead to useful conclusions. In particular, we find that the standard convergence results for value based approaches also apply to the dual case, even in the presence of function approximation and off-policy updating. The dual approach appears to hold an advantage over the standard primal view of DP/RL in one major sense: since the fundamental objects being represented are normalized probability distributions (i.e., belong to a bounded simplex), dual updates cannot diverge. In particular, we find that dual updates converge (i.e. avoid oscillation) in the very circumstance where primal updates can and often do diverge: gradient-based off-policy updates with linear function approximation [5, 6].

## 2   Preliminaries

We consider the problem of computing an optimal behavior strategy in a *Markov decision process* (MDP), defined by a set of actions $A$, a set of states $S$, a $|S||A|$ by $|S|$ transition matrix $P$, a reward vector $\mathbf{r}$ and a discount factor $\gamma$, where we assume the goal is to maximize the infinite horizon *discounted* reward $r_0 + \gamma r_1 + \gamma^2 r_2 + \cdots = \sum_{t=0}^{\infty} \gamma^t r_t$. It is known that an optimal behavior strategy can always be expressed by a stationary *policy*, whose entries $\boldsymbol{\pi}_{(sa)}$ specify the probability of taking action $a$ in state $s$. Below, we represent a policy $\boldsymbol{\pi}$ by an equivalent representation as an $|S| \times |S||A|$ matrix $\Pi$ where $\Pi_{(s,s'a)} = \boldsymbol{\pi}_{(sa)}$ if $s' = s$, otherwise 0. One can quickly verify that the matrix product $\Pi P$ gives the *state-to-state* transition probabilities induced by the policy $\boldsymbol{\pi}$ in the environment $P$, and that $P\Pi$ gives the *state-action to state-action* transition probabilities induced by policy $\boldsymbol{\pi}$ in $P$. The problem is to compute an optimal policy given either (a) a complete

---

[*]Current affiliation: Computer Sciences Laboratory, Australian National University, tao.wang@anu.edu.au.

specification of the environmental variables $P$ and $\mathbf{r}$ (the "*planning problem*"), or (b) limited access to the environment through observed states and rewards and the ability to select actions to cause further state transitions (the "*learning problem*"). The first problem is normally tackled by LP or DP methods, and the second by RL methods. In this paper, we will restrict our attention to scenario (a).

## 3   Dual Representations

Traditionally, DP methods for solving the MDP planning problem are typically expressed in terms of the primal value function. However, [4] demonstrated that all the classical algorithms have natural duals expressed in terms of state and state-action probability distributions.

In the primal representation, the policy state-action value function can be specified by an $|S||A| \times 1$ vector $\mathbf{q} = \sum_{i=0}^{\infty} \gamma^i (P\Pi)^i \mathbf{r}$ which satisfies $\mathbf{q} = \mathbf{r} + \gamma P\Pi\mathbf{q}$. To develop a *dual* form of state-action policy evaluation, one considers the linear system $\mathbf{d}^\top = (1-\gamma)\boldsymbol{\nu}^\top + \gamma \mathbf{d}^\top P\Pi$, where $\boldsymbol{\nu}$ is the initial distribution over state-action pairs. Not only is $\mathbf{d}$ a proper probability distribution over state-action pairs, it also allows one to easily compute the expected discounted return of the policy $\boldsymbol{\pi}$. However, recovering the state-action distribution $\mathbf{d}$ is inadequate for *policy improvement*. Therefore, one considers the following $|S||A| \times |S||A|$ matrix $H = (1-\gamma)I + \gamma P\Pi H$. The matrix $H$ that satisfies this linear relation is similar to $\mathbf{d}^\top$, in that each row is a probability distribution and the entries $H_{(sa,s'a')}$ correspond to the probability of discounted state-action visits to $(s'a')$ for a policy $\boldsymbol{\pi}$ starting in state-action pair $(sa)$. Unlike $\mathbf{d}^\top$, however, $H$ drops the dependence on $\boldsymbol{\nu}$, giving $(1-\gamma)\mathbf{q} = H\mathbf{r}$. That is, given $H$ we can easily recover the state-action values of $\boldsymbol{\pi}$.

For policy improvement, in the primal representation one can derive an improved policy $\boldsymbol{\pi}'$ via the update $a^*(s) = \arg\max_a \mathbf{q}_{(sa)}$ and $\boldsymbol{\pi}'_{(sa)} = 1$ if $a = a^*(s)$, otherwise 0. The dual form of the policy update can be expressed in terms of the state-action matrix $H$ for $\boldsymbol{\pi}$ is $a^*(s) = \arg\max_a H_{(sa,:)}\mathbf{r}$. In fact, since $(1-\gamma)\mathbf{q} = H\mathbf{r}$, the two policy updates given in the primal and dual respectively, must lead to the same resulting policy $\boldsymbol{\pi}'$. Further details are given in [4].

## 4   DP algorithms and convergence

We first investigate whether dynamic programming operators with the dual representations exhibit the same (or better) convergence properties to their primal counterparts. These questions will be answered in the affirmative. In the tabular case, dynamic programming algorithms can be expressed by operators that are successively applied to current approximations (vectors in the primal case, matrices in the dual), to bring them closer to a target solution; namely, the fixed point of a desired Bellman equation. Consider two standard operators, the on-policy update and the max-policy update.

For a given policy $\Pi$, the on-policy operator $\mathcal{O}$ is defined as
$$\mathcal{O}\mathbf{q} = \mathbf{r} + \gamma P\Pi\mathbf{q} \qquad \text{and} \qquad \mathcal{O}H = (1-\gamma)I + \gamma P\Pi H,$$
for the primal and dual cases respectively. The goal of the on-policy update is to bring current representations closer to satisfying the policy-specific Bellman equations,
$$\mathbf{q} = \mathbf{r} + \gamma P\Pi\mathbf{q} \qquad \text{and} \qquad H = (1-\gamma)I + \gamma P\Pi H$$
The max-policy operator $\mathcal{M}$ is different in that it is neither linear nor defined by any reference policy, but instead applies a greedy max update to the current approximations
$$\mathcal{M}\mathbf{q} = \mathbf{r} + \gamma P\Pi^*[\mathbf{q}] \qquad \text{and} \qquad \mathcal{M}H = (1-\gamma)I + \gamma P\Pi_r^*[H],$$
where $\Pi^*[\mathbf{q}]_{(s)} = \max_a \mathbf{q}_{(sa)}$ and $\Pi_r^*[H]_{(s,:)} = H_{(sa'(s),:)}$ such that $a'(s) = \arg\max_a [H\mathbf{r}]_{(sa)}$. The goal of this greedy update is to bring the representations closer to satisfying the optimal-policy Bellman equations $\mathbf{q} = \mathbf{r} + \gamma P\Pi^*[\mathbf{q}]$ and $H = (1-\gamma)I + \gamma P\Pi_r^*[H]$.

### 4.1   On-policy convergence

For the on-policy operator $\mathcal{O}$, convergence to the Bellman fixed point is easily proved in the primal case, by establishing a contraction property of $\mathcal{O}$ with respect to a specific norm on $\mathbf{q}$ vectors. In particular, one defines a weighted 2-norm with weights given by the stationary distribution determined by the policy $\Pi$ and transition model $P$: Let $\mathbf{z} \geq 0$ be a vector such that $\mathbf{z}^\top P\Pi = \mathbf{z}^\top$; that is, $\mathbf{z}$ is the stationary state-action visit distribution for $P\Pi$. Then the norm is defined as

$\|\mathbf{q}\|_{\mathbf{z}}^2 = \mathbf{q}^\top Z\mathbf{q} = \sum_{(sa)} \mathbf{z}_{(sa)}\mathbf{q}_{(sa)}^2$, where $Z = \mathrm{diag}(\mathbf{z})$. It can be shown that $\|P\Pi\mathbf{q}\|_{\mathbf{z}} \leq \|\mathbf{q}\|_{\mathbf{z}}$ and $\|\mathcal{O}\mathbf{q}_1 - \mathcal{O}\mathbf{q}_2\|_{\mathbf{z}} \leq \gamma\|\mathbf{q}_1 - \mathbf{q}_2\|_{\mathbf{z}}$ (see [7]). Crucially, for this norm, a state-action transition is not an expansion [7]. By the contraction map fixed point theorem [2] there exists a unique fixed point of $\mathcal{O}$ in the space of vectors $\mathbf{q}$. Therefore, repeated applications of the on-policy operator converge to a vector $\mathbf{q}_\Pi$ such that $\mathbf{q}_\Pi = \mathcal{O}\mathbf{q}_\Pi$; that is, $\mathbf{q}_\Pi$ satisfies the policy based Bellman equation.

Analogously, for the dual representation $H$, one can establish convergence of the on-policy operator by first defining an approximate weighted norm over matrices and then verifying that $\mathcal{O}$ is a contraction with respect to this norm. Define

$$\|H\|_{\mathbf{z},\mathbf{r}}^2 \;=\; \|H\mathbf{r}\|_{\mathbf{z}}^2 \;=\; \sum_{(sa)} \mathbf{z}_{(sa)}\Big( \sum_{(s'a')} H_{(sa,s'a')}\mathbf{r}_{(s'a')} \Big)^2 \tag{1}$$

It is easily verified that this definition satisfies the property of a pseudo-norm, and in particular, satisfies the triangle inequality. This weighted 2-norm is defined with respect to the stationary distribution $\mathbf{z}$, but also the reward vector $\mathbf{r}$. Thus, the magnitude of a row normalized matrix is determined by the magnitude of the weighted reward expectations it induces.

Interestingly, this definition allows us to establish the same non-expansion and contraction results as the primal case. We can have $\|P\Pi H\|_{\mathbf{z},\mathbf{r}} \leq \|H\|_{\mathbf{z},\mathbf{r}}$ by arguments similar to the primal case. Moreover, the on-policy operator is a contraction with respect to $\|\cdot\|_{\mathbf{z},\mathbf{r}}$.

**Lemma 1** $\|\mathcal{O}H_1 - \mathcal{O}H_2\|_{\mathbf{z},\mathbf{r}} \leq \gamma\|H_1 - H_2\|_{\mathbf{z},\mathbf{r}}$

*Proof:* $\|\mathcal{O}H_1 - \mathcal{O}H_2\|_{\mathbf{z},\mathbf{r}} \;=\; \gamma\|P\Pi(H_1 - H_2)\|_{\mathbf{z},\mathbf{r}} \;\leq\; \gamma\|H_1 - H_2\|_{\mathbf{z},\mathbf{r}}$ since $\|P\Pi H\|_{\mathbf{z},\mathbf{r}} \leq \|H\|_{\mathbf{z},\mathbf{r}}$. ∎

Thus, once again by the contraction map fixed point theorem there exists a fixed point of $\mathcal{O}$ among row normalized matrices $H$, and repeated applications of $\mathcal{O}$ will converge to a matrix $H_\Pi$ such that $\mathcal{O}H_\Pi = H_\Pi$; that is, $H_\Pi$ satisfies the policy based Bellman equation for dual representations. This argument shows that on-policy dynamic programming converges in the dual representation, without making direct reference to the primal case. We will use these results below.

## 4.2 Max-policy convergence

The strategy for establishing convergence for the nonlinear max operator is similar to the on-policy case, but involves working with a different norm. Instead of considering a 2-norm weighted by the visit probabilities induced by a fixed policy, one simply uses the max-norm in this case: $\|\mathbf{q}\|_\infty = \max_{(sa)} |q_{(sa)}|$. The contraction property of the $\mathcal{M}$ operator with respect to this norm can then be easily established in the primal case: $\|\mathcal{M}\mathbf{q}_1 - \mathcal{M}\mathbf{q}_2\|_\infty \leq \gamma\|\mathbf{q}_1 - \mathbf{q}_2\|_\infty$ (see [2]). As in the on-policy case, contraction suffices to establish the existence of a unique fixed point of $\mathcal{M}$ among vectors $\mathbf{q}$, and that repeated application of $\mathcal{M}$ converges to this fixed point $\mathbf{q}_*$ such that $\mathcal{M}\mathbf{q}_* = \mathbf{q}_*$.

To establish convergence of the off-policy update in the dual representation, first define the max-norm for state-action visit distribution as

$$\|H\|_\infty \;=\; \max_{(sa)} \Big| \sum_{(s'a')} H_{(sa,s'a')}\mathbf{r}_{(s'a')} \Big| \tag{2}$$

Then one can simply reduce the dual to the primal case by appealing to the relationship $(1-\gamma)\mathcal{M}\mathbf{q} = \mathcal{M}H\mathbf{r}$ to prove convergence of $\mathcal{M}H$.

**Lemma 2** *If* $(1-\gamma)\mathbf{q} = H\mathbf{r}$, *then* $(1-\gamma)\mathcal{M}\mathbf{q} = \mathcal{M}H\mathbf{r}$.
*Proof:* $(1-\gamma)\mathcal{M}\mathbf{q} = (1-\gamma)\mathbf{r}+\gamma P\Pi^*[(1-\gamma)\mathbf{q}]) = (1-\gamma)\mathbf{r}+\gamma P\Pi^*[H\mathbf{r}] = (1-\gamma)\mathbf{r}+\gamma P\Pi^*_{\mathbf{r}}[H]\mathbf{r} = \mathcal{M}H\mathbf{r}$ where the second equality holds since we assumed $(1-\gamma)\mathbf{q}_{(sa)} = [H\mathbf{r}]_{(sa)}$ for all $(sa)$. ∎

Thus, given convergence of $\mathcal{M}\mathbf{q}$ to a fixed point $\mathcal{M}\mathbf{q}_* = \mathbf{q}_*$, the same must also hold for $\mathcal{M}H$. However, one subtlety here is that the dual fixed point is not unique. This is not a contradiction because the norm on dual representations $\|\cdot\|_{\mathbf{z},\mathbf{r}}$ is in fact just a pseudo-norm, not a proper norm. That is, the relationship between $H$ and $\mathbf{q}$ is many to one, and several matrices can correspond to the same $\mathbf{q}$. These matrices form a convex subspace (in fact, a simplex), since if $H_1\mathbf{r} = (1-\gamma)\mathbf{q}$ and $H_2\mathbf{r} = (1-\gamma)\mathbf{q}$ then $(\alpha H_1 + (1-\alpha)H_2)\mathbf{r} = (1-\gamma)\mathbf{q}$ for any $\alpha$, where furthermore $\alpha$ must be restricted to $0 \leq \alpha \leq 1$ to maintain nonnegativity. The simplex of fixed points $\{H_* : \mathcal{M}H_* = H_*\}$ is given by matrices $H_*$ that satisfy $H_*\mathbf{r} = (1-\gamma)\mathbf{q}_*$.

# 5 DP with function approximation

Primal and dual updates exhibit strong equivalence in the tabular case, as they should. However, when we begin to consider approximation, differences emerge. We next consider the convergence properties of the dynamic programming operators in the context of linear basis approximation. We focus on the on-policy case here, because, famously, the max operator does not always have a fixed point when combined with approximation in the primal case [8], and consequently suffers the risk of divergence [5, 6].

Note that the max operator cannot diverge in the dual case, even with basis approximation, by boundedness alone; although the question of whether max updates always converge in this case remains open. Here we establish that a similar bound on approximation error in the primal case can be proved for the dual approach with respect to the on-policy operator.

In the primal case, linear approximation proceeds by fixing a small set of basis functions, forming a $|S||A| \times k$ matrix $\Phi$, where $k$ is the number of bases. The approximation of $\mathbf{q}$ can be expressed by a linear combination of bases $\hat{\mathbf{q}} = \Phi\mathbf{w}$ where $\mathbf{w}$ is a $k \times 1$ vector of adjustable weights. This is equivalent to maintaining the constraint that $\hat{\mathbf{q}} \in \mathrm{col\_span}(\Phi)$. In the dual, a linear approximation to $H$ can be expressed as $\mathrm{vec}(\hat{H}) = \Psi\mathbf{w}$, where the $\mathrm{vec}$ operator creates a column vector from a matrix by stacking the column vectors of the matrix below one another, $\mathbf{w}$ is a $k \times 1$ vector of adjustable weights as it is in the primal case, and $\Psi$ is a $(|S||A|)^2 \times k$ matrix of basis functions. To ensure that $\hat{H}$ remains a nonnegative, row normalized approximation to $H$, we simply add the constraints that $\hat{H} \in \mathrm{simplex}(\Psi) \equiv \{\hat{H} : \mathrm{vec}(\hat{H}) = \Psi\mathbf{w}, \Psi \geq 0, (\mathbf{1}^\top \otimes I)\Psi = \mathbf{1}\mathbf{1}^\top, \mathbf{w} \geq 0, \mathbf{w}^\top \mathbf{1} = \mathbf{1}\}$ where the operator $\otimes$ is the Kronecker product.

In this section, we first introduce operators (projection and gradient step operators) that ensure the approximations stay representable in the given basis. Then we consider their composition with the on-policy and off-policy updates, and analyze their convergence properties. For the composition of the on-policy update and projection operators, we establish a similar bound on approximation error in the dual case as in the primal case.

## 5.1 Projection Operator

Recall that in the primal, the action value function $\mathbf{q}$ is approximated by a linear combination of bases in $\Phi$. Unfortunately, there is no reason to expect $\mathcal{O}\mathbf{q}$ or $\mathcal{M}\mathbf{q}$ to stay in the column span of $\Phi$, so a best approximation is required. The subtlety resolved by Tsitsiklis and Van Roy [7] is to identify a particular form of best approximation—weighted least squares—that ensures convergence is still achieved when combined with the on-policy operator $\mathcal{O}$. Unfortunately, the fixed point of this combined update operator is not guaranteed to be the best representable approximation of $\mathcal{O}$'s fixed point, $\mathbf{q}_\Pi$. Nevertheless, a bound can be proved on how close this altered fixed point is to the best representable approximation.

We summarize a few details that will be useful below: First, the best least squares approximation is computed with respect to the distribution $\mathbf{z}$. The map from a general $\mathbf{q}$ vector onto its best approximation in $\mathrm{col\_span}(\Phi)$ is defined by another operator, $\mathcal{P}$, which projects $\mathbf{q}$ into the column span of $\Phi$, $\mathcal{P}\mathbf{q} = \mathrm{argmin}_{\hat{\mathbf{q}} \in \mathrm{col\_span}(\Phi)} \|\mathbf{q} - \hat{\mathbf{q}}\|_{\mathbf{z}}^2 = \Phi(\Phi^\top Z\Phi)^{-1}\Phi^\top Z\mathbf{q}$, where $\hat{\mathbf{q}}$ is an approximation for value function $\mathbf{q}$. The important property of this weighted projection is that it is a non-expansion operator in $\|\cdot\|_{\mathbf{z}}$, i.e., $\|\mathcal{P}\mathbf{q}\|_{\mathbf{z}} \leq \|\mathbf{q}\|_{\mathbf{z}}$, which can be easily obtained from the generalized Pythagorean theorem. Approximate dynamic programming then proceeds by composing the two operators—the on-policy update $\mathcal{O}$ with the subspace projection $\mathcal{P}$—to compute the best representable approximation of the one step update. This combined operator is guaranteed to converge, since composing a non-expansion with a contraction is still a contraction, i.e., $\|\mathbf{q}_+ - \mathbf{q}_\Pi\|_{\mathbf{z}} \leq \frac{1}{1-\gamma}\|\mathbf{q}_\Pi - \mathcal{P}\mathbf{q}_\Pi\|_{\mathbf{z}}$ [7].

Linear function approximation in the dual case is a bit more complicated because matrices are being represented, not vectors, and moreover the matrices need to satisfy row normalization and nonnegativity constraints. Nevertheless, a very similar approach to the primal case can be successfully applied. Recall that in the dual, the state-action visit distribution $H$ is approximated by a linear combination of bases in $\Psi$. As in the primal case, there is no reason to expect that an update like $\mathcal{O}H$ should keep the matrix in the simplex. Therefore, a projection operator must be constructed that determines the best representable approximation to $\mathcal{O}H$. One needs to be careful to define

this projection with respect to the right norm to ensure convergence. Here, the pseudo-norm $\|\cdot\|_{\mathbf{z},\mathbf{r}}$ defined in Equation 1 suits this purpose. Define the weighted projection operator $\mathcal{P}$ over matrices

$$\mathcal{P}H = \operatorname*{argmin}_{\hat{H} \in \text{simplex}(\Psi)} \|H - \hat{H}\|_{\mathbf{z},\mathbf{r}}^2 \qquad (3)$$

The projection could be obtained by solving the above quadratic program. A key result is that this projection operator is a non-expansion with respect to the pseudo-norm $\|\cdot\|_{\mathbf{z},\mathbf{r}}$.

**Theorem 1** $\|\mathcal{P}H\|_{\mathbf{z},\mathbf{r}} \leq \|H\|_{\mathbf{z},\mathbf{r}}$

*Proof:* The easiest way to prove the theorem is to observe that the projection operator $\mathcal{P}$ is really a composition of three orthogonal projections: first, onto the linear subspace $\text{span}(\Psi)$, then onto the subspace of row normalized matrices $\text{span}(\Psi) \cap \{H : H\mathbf{1} = \mathbf{1}\}$, and finally onto the space of nonnegative matrices $\text{span}(\Psi) \cap \{H : H\mathbf{1} = \mathbf{1}\} \cap \{H : H \geq 0\}$. Note that the last projection into the nonnegative halfspace is equivalent to a projection into a linear subspace for some hyperplane tangent to the simplex. Each one of these projections is a non-expansion in $\|\cdot\|_{\mathbf{z},\mathbf{r}}$ in the same way: a generalized Pythagorean theorem holds. Consider just one of these linear projections $\mathcal{P}_1$

$$\begin{aligned}
\|H\|_{\mathbf{z},\mathbf{r}}^2 &= \|\mathcal{P}_1 H + H - \mathcal{P}_1 H\|_{\mathbf{z},\mathbf{r}}^2 = \|\mathcal{P}_1 H\mathbf{r} + H\mathbf{r} - \mathcal{P}_1 H\mathbf{r}\|_{\mathbf{z}}^2 \\
&= \|\mathcal{P}_1 H\mathbf{r}\|_{\mathbf{z}}^2 + \|H\mathbf{r} - \mathcal{P}_1 H\mathbf{r}\|_{\mathbf{z}}^2 = \|\mathcal{P}_1 H\|_{\mathbf{z},\mathbf{r}}^2 + \|H - \mathcal{P}_1 H\|_{\mathbf{z},\mathbf{r}}^2
\end{aligned}$$

Since the overall projection is just a composition of non-expansions, it must be a non-expansion. ∎

As in the primal, approximate dynamic programming can be implemented by composing the on-policy update $\mathcal{O}$ with the projection operator $\mathcal{P}$. Since $\mathcal{O}$ is a contraction and $\mathcal{P}$ a non-expansion, $\mathcal{P}\mathcal{O}$ must also be a contraction, and it then follows that it has a fixed point. Note that, as in the tabular case, this fixed point is only unique up to $H\mathbf{r}$-equivalence, since the pseudo-norm $\|\cdot\|_{\mathbf{z},\mathbf{r}}$ does not distinguish $H_1$ and $H_2$ such that $H_1\mathbf{r} = H_2\mathbf{r}$. Here too, the fixed point is actually a simplex of equivalent solutions. For simplicity, we denote the simplex of fixed points for $\mathcal{P}\mathcal{O}$ by some representative $H_+ = \mathcal{P}\mathcal{O}H_+$. Finally, we can recover an approximation bound that is analogous to the primal bound, which bounds the approximation error between $H_+$ and the best representable approximation to the on-policy fixed point $H_\Pi = \mathcal{O}H_\Pi$.

**Theorem 2** $\|H_+ - H_\Pi\|_{\mathbf{z},\mathbf{r}} \leq \frac{1}{1-\gamma}\|\mathcal{P}H_\Pi - H_\Pi\|_{\mathbf{z},\mathbf{r}}$

*Proof:* First note that $\|H_+ - H_\Pi\|_{\mathbf{z},\mathbf{r}} = \|H_+ - \mathcal{P}H_\Pi + \mathcal{P}H_\Pi - H_\Pi\|_{\mathbf{z},\mathbf{r}} \leq \|H_+ - \mathcal{P}H_\Pi\|_{\mathbf{z},\mathbf{r}} + \|\mathcal{P}H_\Pi - H_\Pi\|_{\mathbf{z},\mathbf{r}}$ by generalized Pythagorean theorem. Then since $H_+ = \mathcal{P}\mathcal{O}H_+$ and $\mathcal{P}$ is a non-expansion operator, we have $\|H_+ - \mathcal{P}H_\Pi\|_{\mathbf{z},\mathbf{r}} = \|\mathcal{P}\mathcal{O}H_+ - \mathcal{P}H_\Pi\|_{\mathbf{z},\mathbf{r}} \leq \|\mathcal{O}H_+ - H_\Pi\|_{\mathbf{z},\mathbf{r}}$. Finally, using $H_\Pi = \mathcal{O}H_\Pi$ and Lemma 1, we obtain $\|\mathcal{O}H_+ - H_\Pi\|_{\mathbf{z},\mathbf{r}} = \|\mathcal{O}H_+ - \mathcal{O}H_\Pi\|_{\mathbf{z},\mathbf{r}} \leq \gamma\|H_+ - H_\Pi\|_{\mathbf{z},\mathbf{r}}$. Thus $(1-\gamma)\|H_+ - H_\Pi\|_{\mathbf{z},\mathbf{r}} \leq \|\mathcal{P}H_\Pi - H_\Pi\|_{\mathbf{z},\mathbf{r}}$. ∎

To compare the primal and dual results, note that despite the similarity of the bounds, the projection operators do not preserve the tight relationship between primal and dual updates. That is, even if $(1-\gamma)\mathbf{q} = H\mathbf{r}$ and $(1-\gamma)(\mathcal{O}\mathbf{q}) = (\mathcal{O}H)\mathbf{r}$, it is not true in general that $(1-\gamma)(\mathcal{P}\mathcal{O}\mathbf{q}) = (\mathcal{P}\mathcal{O}H)\mathbf{r}$. The most obvious difference comes from the fact that in the dual, the space of $H$ matrices has bounded diameter, whereas in the primal, the space of $\mathbf{q}$ vectors has unbounded diameter in the natural norms. Automatically, the dual updates cannot diverge with compositions like $\mathcal{P}\mathcal{O}$ and $\mathcal{P}\mathcal{M}$; yet, in the primal case, the update $\mathcal{P}\mathcal{M}$ is known to not have fixed points in some circumstances [8].

## 5.2 Gradient Operator

In large scale problems one does not normally have the luxury of computing full dynamic programming updates that evaluate complete expectations over the entire domain, since this requires knowing the stationary visit distribution $\mathbf{z}$ for $P\Pi$ (essentially requiring one to know the model of the MDP). Moreover, full least squares projections are usually not practical to compute. A key intermediate step toward practical DP and RL algorithms is to formulate gradient step operators that only approximate full projections. Conveniently, the gradient update and projection operators are independent of the on-policy and off-policy updates and can be applied in either case. However, as we will see below, the gradient update operator causes significant instability in the off-policy update,

to the degree that divergence is a common phenomenon (much more so than with full projections). Composing approximation with an off-policy update (max operator) in the primal case can be very dangerous. All other operator combinations are better behaved in practice, and even those that are not known to converge usually behave reasonably. Unfortunately, composing the gradient step with an off-policy update is a common algorithm attempted in reinforcement learning (Q-learning with function approximation), despite being the most unstable.

In the dual representation, one can derive a gradient update operator in a similar way to the primal, except that it is important to maintain the constraints on the parameters $\mathbf{w}$, since the basis functions are probability distributions. We start by considering the projection objective

$$J_H \;=\; \frac{1}{2}\|H - \hat{H}\|_{\mathbf{z},\mathbf{r}}^2 \text{ subject to } \quad \text{vec}(\hat{H}) = \Psi\mathbf{w}, \;\; \mathbf{w} \geq 0, \;\; \mathbf{w}^\top\mathbf{1} = 1$$

The unconstrained gradient of the above objective with respect to $\mathbf{w}$ is

$$\nabla_{\mathbf{w}} J_H \;=\; \Psi^\top(\mathbf{r}^\top\!\otimes I)^\top Z(\mathbf{r}^\top\!\otimes I)(\Psi\mathbf{w} - h) \;=\; \Gamma^\top Z(\mathbf{r}^\top\!\otimes I)(\hat{\mathbf{h}} - \mathbf{h})$$

where $\Gamma = (\mathbf{r}^\top \otimes I)\Psi$, $\mathbf{h} = \text{vec}(H)$, and $\hat{\mathbf{h}} = \text{vec}(\hat{H})$. However, this gradient step cannot be followed directly because we need to maintain the constraints. The constraint $\mathbf{w}^\top\mathbf{1} = 1$ can be maintained by first projecting the gradient onto it, obtaining $\delta\mathbf{w} = (I - \frac{1}{k}\mathbf{1}\mathbf{1}^\top)\nabla_{\mathbf{w}} J_H$. Thus, the weight vector can be updated by

$$\mathbf{w}_{t+1} \;=\; \mathbf{w}_t - \alpha\delta\mathbf{w} \;=\; \mathbf{w}_t - \alpha(I - \frac{1}{k}\mathbf{1}\mathbf{1}^\top)\Gamma^\top Z(\mathbf{r}^\top \otimes I)(\hat{\mathbf{h}} - \mathbf{h})$$

where $\alpha$ is a step-size parameter. Then the gradient operator can then be defined by

$$\mathcal{G}_{\hat{\mathbf{h}}}\mathbf{h} \;=\; \hat{\mathbf{h}} - \alpha\Psi\delta\mathbf{w} \;=\; \hat{\mathbf{h}} - \alpha\Psi(I - \frac{1}{k}\mathbf{1}\mathbf{1}^\top)\Gamma^\top Z(\mathbf{r}^\top \otimes I)(\hat{\mathbf{h}} - \mathbf{h})$$

(Note that to further respect the box constraints, $0 \leq \mathbf{h} \leq 1$, the stepsize might need to be reduced and additional equality constraints might have to be imposed on some of the components of $\mathbf{h}$ that are at the boundary values.)

Similarly as in the primal, since the target vector $H$ (i.e., $\mathbf{h}$) is determined by the underlying dynamic programming update, this gives the composed updates

$$\mathcal{G}\mathcal{O}\hat{\mathbf{h}} \;=\; \hat{\mathbf{h}} - \alpha\Psi(I - \frac{1}{k}\mathbf{1}\mathbf{1}^\top)\Gamma^\top Z(\mathbf{r}^\top\!\otimes I)(\hat{\mathbf{h}} - \mathcal{O}\hat{\mathbf{h}}) \quad \text{and}$$

$$\mathcal{G}\mathcal{M}\hat{\mathbf{h}} \;=\; \hat{\mathbf{h}} - \alpha\Psi(I - \frac{1}{k}\mathbf{1}\mathbf{1}^\top)\Gamma^\top(\mathbf{r}^\top\!\otimes I)(\hat{\mathbf{h}} - \mathcal{M}\hat{\mathbf{h}})$$

respectively for the on-policy and off-policy cases (ignoring the additional equality constraints).

Thus far, the dual approach appears to hold an advantage over the standard primal approach, since convergence holds in every circumstance where the primal updates converge, and yet the dual updates are guaranteed never to diverge because the fundamental objects being represented are normalized probability distributions (i.e., belong to a bounded simplex). We now investigate the convergence properties of the various updates empirically.

## 6   Experimental Results

To investigate the effectiveness of the dual representations, we conducted experiments on various domains, including randomly synthesized MDPs, Baird's *star problem* [5], and on the *mountain car* problem. The randomly synthesized MDP domains allow us to test the general properties of the algorithms. The star problem is perhaps the most-cited example of a problem where Q-learning with linear function approximation diverges [5], and the mountain car domain has been prone to divergence with some primal representations [9] although successful results were reported when bases are selected by sparse tile coding [10].

For each problem domain, twelve algorithms were run over 100 repeats with a horizon of 1000 steps. The algorithms were: tabular on-policy ($\mathcal{O}$), projection on-policy ($\mathcal{P}\mathcal{O}$), gradient on-policy ($\mathcal{G}\mathcal{O}$), tabular off-policy ($\mathcal{M}$), projection off-policy ($\mathcal{P}\mathcal{M}$), and gradient off-policy ($\mathcal{G}\mathcal{M}$), for both the primal and the dual. The discount factor was set to $\gamma = 0.9$. For on-policy algorithms, we measure the difference between the values generated by the algorithms and those generated by the analytically determined fixed-point. For off-policy algorithms, we measure the difference between the values generated by the resulting policy and the values of the optimal policy. The step size for the gradient updates was 0.1 for primal representations and 100 for dual representations. The initial values of

state-action value functions $\mathbf{q}$ are set according to the standard normal distribution, and state-action visit distributions $H$ are chosen uniformly randomly with row normalization. Since the goal is to investigate the convergence of the algorithms without carefully crafting features, we also choose random basis functions according to a standard normal distribution for the primal representations, and random basis distributions according to a uniform distribution for the dual representations.

**Randomly Synthesized MDPs.** For the synthesized MDPs, we generated the transition and reward functions of the MDPs randomly—the transition function is uniformly distributed between $0$ and $1$ and the reward function is drawn from a standard normal. Here we only reported the results of random MDPs with $100$ states, $5$ actions, and $10$ bases, observed consistent convergence of the dual representations on a variety of MDPs, with different numbers of states, actions, and bases. In Figure 1(right), the curve for the gradient off-policy update ($\mathcal{GM}$) in the primal case (dotted line with the circle marker) blows up (diverges), while all the other algorithms in Figure 1 converge. Interestingly, the approximate error of the dual algorithm $\mathcal{POH}$ ($4.60 \times 10^{-3}$) is much smaller than the approximate error of the corresponding primal algorithm $\mathcal{POq}$ ($4.23 \times 10^{-2}$), even though their theoretical bounds are the same (see Figure 1(left)).

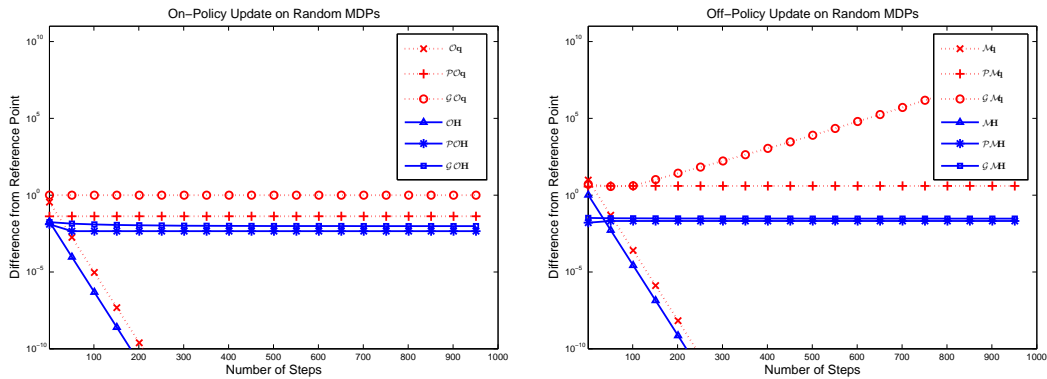

Figure 1: Updates of state-action value $\mathbf{q}$ and visit distribution $H$ on randomly synthesized MDPs

**The Star Problem.** The star problem has $7$ states and $2$ actions. The reward function is zero for each transition. In these experiments, we used the same fixed policy and linear value function approximation as in [5]. In the dual, the number of bases is also set to $14$ and the initial values of the state-action visit distribution matrix $H$ are uniformly distributed random numbers between $0$ and $1$ with row normalization. The gradient off-policy update in the primal case diverges (see the dotted line with the circle marker in Figure 2(right)). However, all the updates with the dual representation algorithms converge.

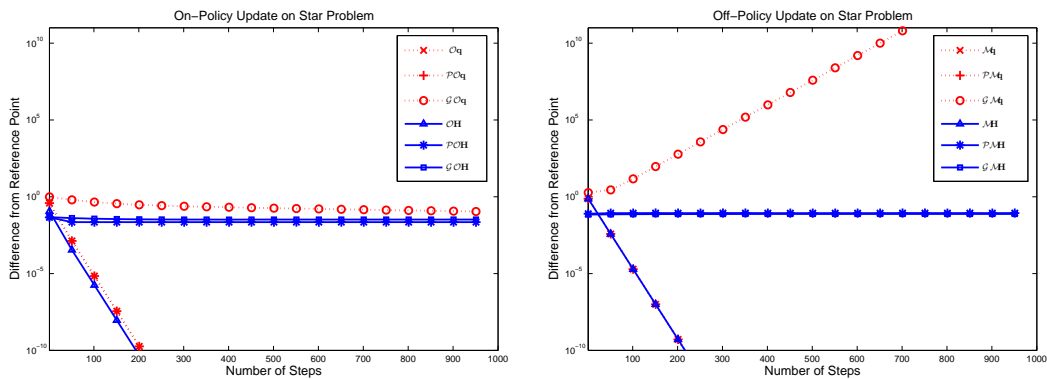

Figure 2: Updates of state-action value $\mathbf{q}$ and visit distribution $H$ on the star problem

**The Mountain Car Problem**   The mountain car domain has continuous state and action spaces, which we discretized with a simple grid, resulting in an MDP with 222 states and 3 actions. The number of bases was chosen to be 5 for both the primal and dual algorithms. For the same reason as before, we chose the bases for the algorithms randomly. In the primal representations with linear function approximation, we randomly generated basis functions according to the standard normal distribution. In the dual representations, we randomly picked the basis distributions according to the uniform distribution. In Figure 3(right), we again observed divergence of the gradient off-policy update on state-action values in the primal, and the convergence of all the dual algorithms (see Figure 3). Again, the approximation error of the projected on-policy update $\mathcal{POH}$ in the dual ($1.90 \times 10^1$) is also considerably smaller than $\mathcal{POq}$ ($3.26 \times 10^2$) in the primal.

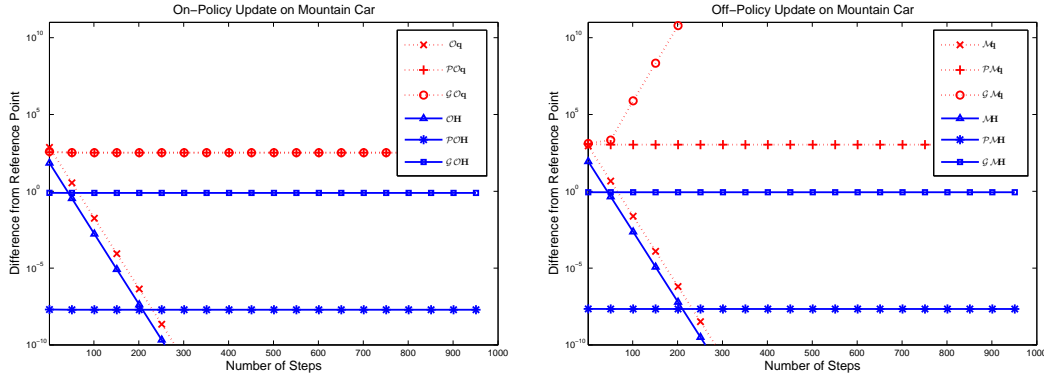

Figure 3: Updates of state-action value $\mathbf{q}$ and visit distribution $H$ on the mountain car problem

# 7   Conclusion

Dual representations maintain an explicit representation of visit distributions as opposed to value functions [4]. We extended the dual dynamic programming algorithms with linear function approximation, and studied the convergence properties of the dual algorithms for planning in MDPs. We demonstrated that dual algorithms, since they are based on estimating normalized probability distributions rather than unbounded value functions, avoid divergence even in the presence of approximation and off-policy updates. Moreover, dual algorithms remain stable in situations where standard value function estimation diverges.

# References

[1]  M. Puterman. *Markov Decision Processes: Discrete Dynamic Programming*. Wiley, 1994.

[2]  D. Bertsekas. *Dynamic Programming and Optimal Control*, volume 2. Athena Scientific, 1995.

[3]  D. Bertsekas and J. Tsitsiklis. *Neuro-Dynamic Programming*. Athena Scientific, 1996.

[4]  T. Wang, M. Bowling, and D. Schuurmans. Dual representations for dynamic programming and reinforcement learning. In *Proceeding of the IEEE International Symposium on ADPRL*, pages 44–51, 2007.

[5]  L. C. Baird. Residual algorithms: Reinforcement learning with function approximation. In *International Conference on Machine Learning*, pages 30–37, 1995.

[6]  R. Sutton and A. Barto. *Reinforcement Learning: An Introduction*. MIT Press, 1998.

[7]  J. Tsitsiklis and B. Van Roy. An analysis of temporal-difference learning with function approximation. *IEEE Trans. Automat. Control*, 42(5):674–690, 1997.

[8]  D. de Farias and B. Van Roy. On the existence of fixed points for approximate value iteration and temporal-difference learning. *J. Optimization Theory and Applic.*, 105(3):589–608, 2000.

[9]  J. A. Boyan and A. W. Moore. Generalization in reinforcement learning: Safely approximating the value function. In *NIPS 7*, pages 369–376, 1995.

[10]  R. S. Sutton. Generalization in reinforcement learning: Successful examples using sparse coarse coding. In *Advances in Neural Information Processing Systems*, pages 1038–1044, 1996.
